# Just One View:
# Invariances in Inferotemporal Cell Tuning

**Maximilian Riesenhuber**          **Tomaso Poggio**
Center for Biological and Computational Learning and
Department of Brain and Cognitive Sciences
Massachusetts Institute of Technology, E25-201
Cambridge, MA 02139
{max,tp}@ai.mit.edu

## Abstract

In macaque inferotemporal cortex (IT), neurons have been found to respond selectively to complex shapes while showing broad tuning ("invariance") with respect to stimulus transformations such as translation and scale changes and a limited tuning to rotation in depth. Training monkeys with novel, paperclip-like objects, Logothetis et al.[9] could investigate whether these invariance properties are due to experience with exhaustively many transformed instances of an object or if there are mechanisms that allow the cells to show response invariance also to previously unseen instances of that object. They found object-selective cells in anterior IT which exhibited limited invariance to various transformations after training with single object views. While previous models accounted for the tuning of the cells for rotations in depth and for their selectivity to a specific object relative to a population of distractor objects,[14,1] the model described here attempts to explain in a biologically plausible way the additional properties of translation and size invariance. Using the same stimuli as in the experiment, we find that model IT neurons exhibit invariance properties which closely parallel those of real neurons. Simulations show that the model is capable of unsupervised learning of view-tuned neurons.

We thank Peter Dayan, Marcus Dill, Shimon Edelman, Nikos Logothetis, Jonathan Murnick and Randy O'Reilly for useful discussions and comments.

# 1 Introduction

Neurons in macaque inferotemporal cortex (IT) have been shown to respond to views of complex objects,[8] such as faces or body parts, even when the retinal image undergoes size changes over several octaves, is translated by several degrees of visual angle[7] or rotated in depth by a certain amount[9] (see [13] for a review).

These findings have prompted researchers to investigate the physiological mechanisms underlying these tuning properties. The original model[14] that led to the physiological experiments of Logothetis *et al.*[9] explains the behavioral view invariance for rotation in depth through the learning and memory of a few example views, each represented by a neuron tuned to that view. Invariant recognition for translation and scale transformations have been explained either as a result of object-specific learning[4] or as a result of a normalization procedure ("shifter") that is applied to any image and hence requires only one object-view for recognition.[12]

A problem with previous experiments has been that they did not illuminate the mechanism underlying invariance since they employed objects (*e.g.,* faces) with which the monkey was quite familiar, having seen them numerous times under various transformations. Recent experiments by Logothetis *et al.*[9] addressed this question by training monkeys to recognize *novel* objects ("paperclips" and amoeba-like objects) with which the monkey had no previous visual experience. After training, responses of IT cells to transformed versions of the training stimuli and to distractors of the same type were collected. Since the views the monkeys were exposed to during training were tightly controlled, the paradigm allowed to estimate the degree of invariance that can be extracted from just one object view.

In particular, Logothetis *et al.*[9] tested the cells' responses to rotations in depth, translation and size changes. Defining "invariance" as yielding a higher response to test views than to distractor objects, they report[9,10] an average rotation invariance over 30°, translation invariance over ±2°, and size invariance of up to ±1 octave around the training view.

These results establish that there are cells showing some degree of invariance even after training with just one object view, thereby arguing against a completely learning-dependent mechanisms that requires visual experience with each transformed instance that is to be recognized. On the other hand, invariance is far from perfect but rather centered around the object views seen during training.

# 2 The Model

Studies of the visual areas in the ventral stream of the macaque visual system[8] show a tendency for cells higher up in the pathway (from V1 over V2 and V4 to anterior and posterior IT) to respond to increasingly complex objects and to show increasing invariance to transformations such as translations, size changes or rotation in depth.[13]

We tried to construct a model that explains the receptive field properties found in the experiment based on a simple feedforward model. Figure 1 shows a cartoon of the model: A retinal input pattern leads to excitation of a set of "V1" cells, in the figure abstracted as having derivative-of-Gaussian receptive field profiles. These "V1" cells are tuned to simple features and have relatively small receptive fields. While they could be cells from a variety of areas, *e.g.,* V1 or V2 (cf. Discussion), for simplicity, we label them as "V1" cells (see figure). Different cells differ in preferred feature, *e.g.,* orientation, preferred spatial frequency (scale), and receptive field location. "V1" cells of the same type (*i.e.,* having the same preferred stimulus, but of different preferred scale and receptive field location) feed into the same neuron in an intermediate layer. These intermediate neurons could be complex cells in V1 or V2 or V4 or even posterior IT: we label them as "V4" cells, in the

same spirit in which we labeled the neurons feeding into them as "V1" units. Thus, a "V4" cell receives inputs from "V1" cells over a large area and different spatial scales ([8] reports an average receptive field size in V4 of 4.4° of visual angle, as opposed to about 1° in V1; for spatial frequency tuning, [3] report an average FWHM of 2.2 octaves, compared to 1.4 (foveally) to 1.8 octaves (parafoveally) in V1[5]). These "V4" cells in turn feed into a layer of "IT" neurons, whose invariance properties are to be compared with the experimentally observed ones.

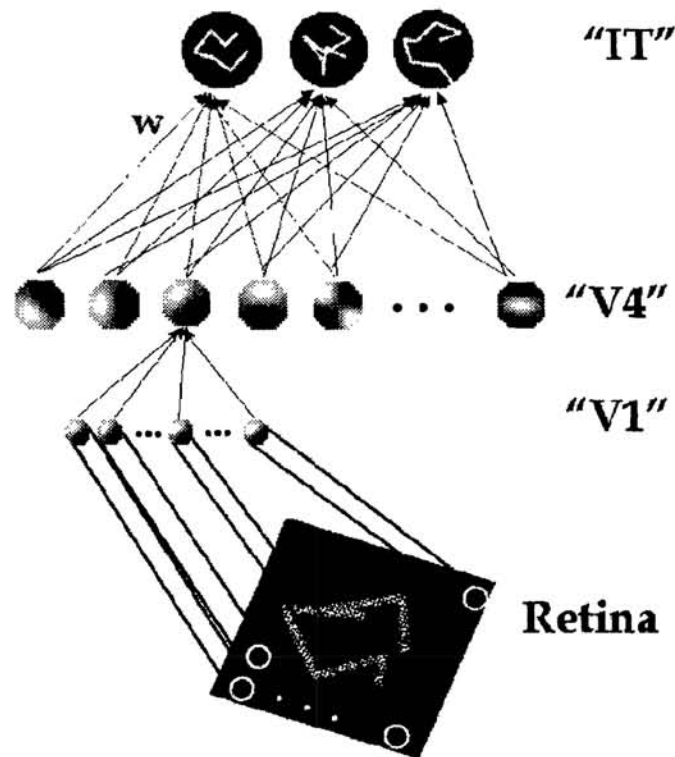

**Figure 1**: Cartoon of the model. See text for explanation.

A crucial element of the model is the mechanism an intermediate neuron uses to pool the activities of its afferents. From the computational point of view, the intermediate neurons should be robust feature detectors, *i.e.,* measure the presence of specific features without being confused by clutter and context in the receptive field. More detailed considerations (Riesenhuber and Poggio, in preparation) show that this cannot be achieved with a response function that just summates over all the afferents (cf. Results). Instead, intermediate neurons in our model perform a "max" operation (akin to a "Winner-Take-All") over all their afferents, *i.e., the response of an intermediate neuron is determined by its most strongly excited afferent.* This hypothesis appears to be compatible with recent data,[15] that show that when two stimuli (gratings of different contrast and orientation) are brought into the receptive field of a V4 cell, the cell's response tends to be close to the stronger of the two individual responses (instead of *e.g.,* the sum as in a linear model).

Thus, the response function $o_i$ of an intermediate neuron $i$ to stimulation with an image $\mathbf{v}$ is

$$o_i = \max_{j \in \mathcal{A}_i} \{ \mathbf{v}_{\alpha(j)} \cdot \xi_j \}, \tag{1}$$

with $\mathcal{A}_i$ the set of afferents to neuron $i$, $\alpha(j)$ the receptive field center of afferent $j$, $\mathbf{v}_{\alpha(j)}$ the (square-normalized) image patch centered at $\alpha(j)$ that corresponds in size to the receptive field, $\xi_j$ (also square-normalized) of afferent $j$ and "$\cdot$" the dot product operation.

Studies have shown that V4 neurons respond to features of "intermediate" complexity such as gratings, corners and crosses.[8] In V4 the receptive fields are comparatively large ($4.4°$ of visual angle on average[8]), while the preferred stimuli are usually much smaller.[3] Interestingly, cells respond independently of the location of the stimulus within the receptive field. Moreover, average V4 receptive field size is comparable to the range of translation invariance of IT cells ($\leq \pm 2°$) observed in the experiment.[9] For afferent receptive fields $\xi_j$, we chose features similar to the ones found for V4 cells in the visual system:[8] bars (modeled as second derivatives of Gaussians) in two orientations, and "corners" of four different orientations and two different degrees of obtuseness. This yielded a total of 10 intermediate neurons. This set of features was chosen to give a compact and biologically plausible representation. Each intermediate cell received input from cells with the same type of preferred stimulus densely covering the visual field of $256 \times 256$ pixels (which thus would correspond to about $4.4°$ of visual angle, the average receptive field size in V4[8]), with receptive field sizes of afferent cells ranging from 7 to 19 pixels in steps of 2 pixels. The features used in this paper represent the first set of features tried, optimizing feature shapes might further improve the model's performance.

The response $t_j$ of top layer neuron $j$ with connecting weights $\mathbf{w}_j$ to the intermediate layer was set to be a Gaussian, centered on $\mathbf{w}_j$,

$$t_j = \frac{1}{\sqrt{2\pi\sigma^2}} \exp \left( -\frac{||\mathbf{o} - \mathbf{w}_j||^2}{2\sigma^2} \right) \tag{2}$$

where $\mathbf{o}$ is the excitation of the intermediate layer and $\sigma$ the variance of the Gaussian, which was chosen based on the distribution of responses (for section 3.1) or learned (for section 3.2).

The stimulus images were views of 21 randomly generated "paperclips" of the type used in the physiology experiment.[9] Distractors were 60 other paperclip images generated by the same method. Training size was $128 \times 128$ pixels.

## 3  Results

### 3.1  Invariance of Representation

In a first set of simulations we investigated whether the proposed model could indeed account for the observed invariance properties. Here we assumed that connection strengths from the intermediate layer cells to the top layer had already been learned by a separate process, allowing us to focus on the tolerance of the representation to the above-mentioned transformations and on the selectivity of the top layer cells.

To establish the tuning properties of view-tuned model neurons, the connections $\mathbf{w}_j$ between the intermediate layer and top layer unit $j$ were set to be equal to the excitation $o_{\text{training}}$ in the intermediate layer caused by the training view. Figure 2 shows the "tuning curve" for rotation in depth and Fig. 3 the response to changes in stimulus size of one such neuron. The neuron shows rotation invariance (*i.e.*, producing a higher response than to any distractor) over about $44°$ and invariance to scale changes over the whole range tested. For translation

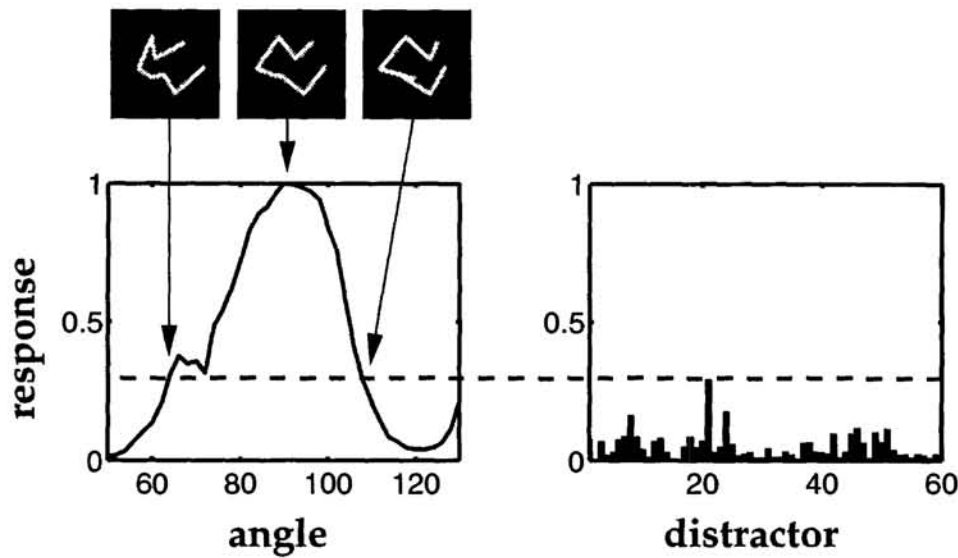

**Figure 2**: Responses of a sample top layer neuron to different views of the training stimulus and to distractors. The left plot shows the rotation tuning curve, with the training view (90° view) shown in the middle image over the plot. The neighboring images show the views of the paperclip at the borders of the rotation tuning curve, which are located where the response to the rotated clip falls below the response to the best distractor (shown in the plot on the right). The neuron exhibits broad rotation tuning over more than 40°.

(not shown), the neuron showed invariance over translations of ±96 pixels around the center in any direction, corresponding to ±1.7° of visual angle.

The average invariance ranges for the 21 tested paperclips were 35° of rotation angle, 2.9 octaves of scale invariance and ±1.8° of translation invariance. Comparing this to the experimentally observed[10] 30°, 2 octaves and ±2°, resp., shows a very good agreement of the invariance properties of model and experimental neurons.

## 3.2 Learning

In the previous section we assumed that the connections from the intermediate layer to a view-tuned neuron in the top layer were pre-set to appropriate values. In this section, we investigate whether the system allows unsupervised learning of view-tuned neurons.

Since biological plausibility of the learning algorithm was not our primary focus here, we chose a general, rather abstract learning algorithm, *viz.* a mixture of Gaussians model trained with the EM algorithm. Our model had four neurons in the top level, the stimuli were views of four paperclips, randomly selected from the 21 paperclips used in the previous experiments. For each clip, the stimulus set contained views from 17 different viewpoints, spanning 34° of viewpoint change. Also, each clip was included at 11 different scales in the stimulus set, covering a range of two octaves of scale change.

Connections $\mathbf{w}_i$ and variances $\sigma_i$, $i = 1, \ldots, 4$, were initialized to random values at the beginning of training. After a few iterations of the EM algorithm (usually less than 30), a stationary state was reached, in which each model neuron had become tuned to views of one paperclip: For each paperclip, all rotated and scaled views were mapped to (*i.e.*, activated most strongly) the same model neuron and views of different paperclips were mapped to different neurons. Hence, when the system is presented with multiple views of different objects, receptive fields of top level neurons self-organize in such a way that different neurons become tuned to different objects.

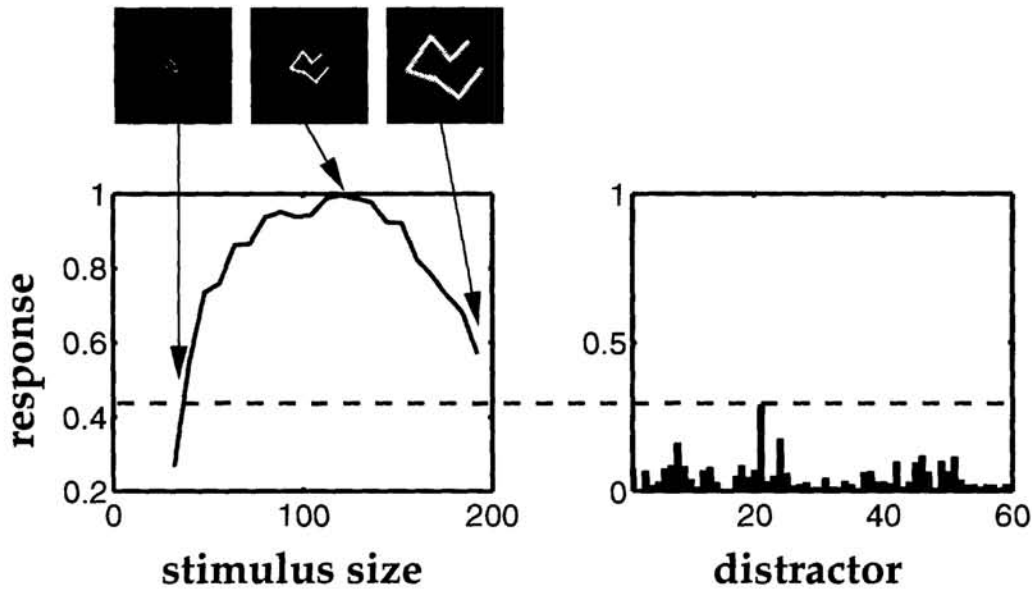

**Figure 3**: Responses of the same top layer neuron as in Fig. 2 to scale changes of the training stimulus and to distractors. The left plot shows the size tuning curve, with the training size (128 × 128 pixels) shown in the middle image over the plot. The neighboring images show scaled versions of the paperclip. Other elements as in Fig. 2. The neuron exhibits scale invariance over more than 2 octaves.

## 4 Discussion

Object recognition is a difficult problem because objects must be recognized irrespective of position, size, viewpoint and illumination. Computational models and engineering implementations have shown that most of the required invariances can be obtained by a relatively simple learning scheme, based on a small set of example views.[14,17] Quite sensibly, the visual system can also achieve some significant degree of scale and translation invariance from just one view. Our simulations show that the maximum response function is a key component in the performance of the model. Without it — *i.e.*, implementing a direct convolution of the filters with the input images and a subsequent summation — invariance to rotation in depth and translation both decrease significantly. Most dramatically, however, invariance to scale changes is abolished completely, due to the strong changes in afferent cell activity with changing stimulus size. Taking the maximum over the afferents, as in our model, always picks the best matching filter and hence produces a more stable response. We expect a maximum mechanism to be essential for recognition-in-context, a more difficult task and much more common than the recognition of isolated objects studied here and in the related psychophysical and physiological experiments.

The recognition of a specific paperclip object is a difficult, subordinate level classification task. It is interesting that our model solves it well and with a performance closely resembling the physiological data on the same task. The model is a more biologically plausible and complete model than previous ones[14,1] but it is still at the level of a plausibility proof rather than a detailed physiological model. It suggests a maximum-like response of intermediate cells as a key mechanism for explaining the properties of view-tuned IT cells, in addition to view-based representations (already described in [1,9]).

Neurons in the intermediate layer currently use a very simple set of features. While this appears to be adequate for the class of paperclip objects, more complex filters might be necessary for more complex stimulus classes like faces. Consequently, future work will aim to improve the filtering step of the model and to test it on more real world stimuli. One can imagine a hierarchy of cell layers, similar to the "S" and "C" layers in Fukushima's

Neocognitron,[6] in which progressively more complex features are synthesized from simple ones. The corner detectors in our model are likely candidates for such a scheme. We are currently investigating the feasibility of such a hierarchy of feature detectors.

The demonstration that unsupervised learning of view-tuned neurons is possible in this representation (which is not clear for related view-based models[14,1]) shows that different views of one object tend to form distinct clusters in the response space of intermediate neurons. The current learning algorithm, however, is not very plausible, and more realistic learning schemes have to be explored, as, for instance, in the attention-based model of Riesenhuber and Dayan[16] which incorporated a learning mechanism using bottom-up and top-down pathways. Combining the two approaches could also demonstrate how invariance over a wide range of transformations can be learned from several example views, as in the case of familiar stimuli. We also plan to simulate detailed physiological implementations of several aspects of the model such as the maximum operation (for instance comparing nonlinear dendritic interactions[11] with recurrent excitation and inhibition). As it is, the model can already be tested in additional physiological experiments, for instance involving partial occlusions.

## References

[1] Bricolo, E, Poggio, T & Logothetis, N (1997). 3D object recognition: A model of view-tuned neurons. In *Advances In Neural Information Processing* 9, 41-47. MIT Press.

[2] Bülthoff, H & Edelman, S (1992). Psychophysical support for a two-dimensional view interpolation theory of object recognition. *Proc. Nat. Acad. Sci. USA* 89, 60-64.

[3] Desimone, R & Schein, S (1987). Visual properties of neurons in area V4 of the macaque: Sensitivity to stimulus form. *J. Neurophys.* 57, 835-868.

[4] Földiák, P (1991). Learning invariance from transformation sequences. *Neural Computation* 3, 194-200.

[5] Foster, KH, Gaska, JP, Nagler, M & Pollen, DA (1985). Spatial and temporal selectivity of neurones in visual cortical areas V1 and V2 of the macaque monkey. *J. Phy.* 365, 331-363.

[6] Fukushima, K (1980). Neocognitron: A self-organizing neural network model for a mechanism of pattern recognition unaffected by shift in position. *Biological Cybernetics* 36, 193-202.

[7] Ito, M, Tamura, H, Fujita, I & Tanaka, K (1995). Size and position invariance of neuronal responses in monkey inferotemporal cortex. *J. Neurophys.* 73, 218–226.

[8] Kobatake, E & Tanaka, K (1995). Neuronal selectivities to complex object features in the ventral visual pathway of the macaque cerebral cortex *J. Neurophys.,* 71, 856-867.

[9] Logothetis, NK, Pauls, J & Poggio, T (1995). Shape representation in the inferior temporal cortex of monkeys. *Current Biology,* 5, 552-563.

[10] Nikos Logothetis, personal communication.

[11] Mel, BW, Ruderman, DL & Archie, KA (1997). Translation-invariant orientation tuning in visual 'complex' cells could derive from intradendritic computations. Manuscript in preparation.

[12] Olshausen, BA, Anderson, CH & Van Essen, DC (1993). A neurobiological model of visual attention and invariant pattern recognition based on dynamic routing of information. *J. Neurosci.* 13, 4700-4719.

[13] Perret, D & Oram, M (1993). Neurophysiology of shape processing. *Image Vision Comput.* 11, 317-333.

[14] Poggio, T & Edelman, S (1990). A Network that learns to recognize 3D objects. *Nature* 343, 263-266.

[15] Reynolds, JH & Desimone, R (1997). Attention and contrast have similar effects on competitive interactions in macaque area V4. *Soc. Neurosc. Abstr.* 23, 302.

[16] Riesenhuber, M & Dayan, P (1997). Neural models for part-whole hierarchies. In *Advances In Neural Information Processing* 9, 17-23. MIT Press.

[17] Ullman, S (1996). *High-level vision: Object recognition and visual cognition.* MIT Press.
